# On a Connection between Kernel PCA and Metric Multidimensional Scaling

**Christopher K. I. Williams**
Division of Informatics
The University of Edinburgh
5 Forrest Hill, Edinburgh EH1 2QL, UK
c.k.i.williams@ed.ac.uk
http://anc.ed.ac.uk

## Abstract

In this paper we show that the kernel PCA algorithm of Schölkopf *et al* (1998) can be interpreted as a form of metric multidimensional scaling (MDS) when the kernel function $k(\mathbf{x}, \mathbf{y})$ is isotropic, i.e. it depends only on $||\mathbf{x} - \mathbf{y}||$. This leads to a metric MDS algorithm where the desired configuration of points is found via the solution of an eigenproblem rather than through the iterative optimization of the stress objective function. The question of kernel choice is also discussed.

## 1   Introduction

Suppose we are given $n$ objects, and for each pair $(i, j)$ we have a measurement of the "dissimilarity" $\delta_{ij}$ between the two objects. In multidimensional scaling (MDS) the aim is to place $n$ points in a low dimensional space (usually Euclidean) so that the interpoint distances $d_{ij}$ have a particular relationship to the original dissimilarities. In *classical scaling* we would like the interpoint distances to be equal to the dissimilarities. For example, classical scaling can be used to reconstruct a map of the locations of some cities given the distances between them.

In *metric MDS* the relationship is of the form $d_{ij} \approx f(\delta_{ij})$ where $f$ is a specific function. In this paper we show that the kernel PCA algorithm of Schölkopf *et al* [7] can be interpreted as performing metric MDS if the kernel function is isotropic. This is achieved by performing classical scaling in the feature space defined by the kernel.

The structure of the remainder of this paper is as follows: In section 2 classical and metric MDS are reviewed, and in section 3 the kernel PCA algorithm is described. The link between the two methods is made in section 4. Section 5 describes approaches to choosing the kernel function, and we finish with a brief discussion in section 6.

## 2 Classical and metric MDS

### 2.1 Classical scaling

Given $n$ objects and the corresponding dissimilarity matrix, classical scaling is an algebraic method for finding a set of points in space so that the dissimilarities are well-approximated by the interpoint distances. The classical scaling algorithm is introduced below by starting with the locations of $n$ points, constructing a dissimilarity matrix based on their Euclidean distances, and then showing how the configuration of the points can be reconstructed (as far as possible) from the dissimilarity matrix.

Let the coordinates of $n$ points in $p$ dimensions be denoted by $\mathbf{x}_i$, $i = 1, \ldots, n$. These can be collected together in a $n \times p$ matrix $X$. The dissimilarities are calculated by $\delta_{ij}^2 = (\mathbf{x}_i - \mathbf{x}_j)^T(\mathbf{x}_i - \mathbf{x}_j)$. Given these dissimilarities, we construct the matrix $A$ such that $a_{ij} = -\frac{1}{2}\delta_{ij}^2$, and then set $B = HAH$, where $H$ is the centering matrix $H = I_n - \frac{1}{n}\mathbf{1}\mathbf{1}^T$. With $\delta_{ij}^2 = (\mathbf{x}_i - \mathbf{x}_j)^T(\mathbf{x}_i - \mathbf{x}_j)$, the construction of $B$ leads to $b_{ij} = (\mathbf{x}_i - \bar{\mathbf{x}})^T(\mathbf{x}_j - \bar{\mathbf{x}})$, where $\bar{\mathbf{x}} = \frac{1}{n}\sum_{i=1}^n \mathbf{x}_i$. In matrix form we have $B = (HX)(HX)^T$, and $B$ is real, symmetric and positive semi-definite. Let the eigendecomposition of $B$ be $B = V\Lambda V^T$, where $\Lambda$ is a diagonal matrix and $V$ is a matrix whose columns are the eigenvectors of $B$. If $p < n$, there will be $n - p$ zero eigenvalues[1]. If the eigenvalues are ordered $\lambda_1 \geq \lambda_2 \geq \cdots \geq \lambda_n \geq 0$, then $B = V_p\Lambda_p V_p^T$, where $\Lambda_p = \text{diag}(\lambda_1, \ldots, \lambda_p)$ and $V_p$ is the $n \times p$ matrix whose columns correspond to the first $p$ eigenvectors of $B$, with the usual normalization so that the eigenvectors have unit length. The matrix $\hat{X}$ of the reconstructed coordinates of the points can be obtained as $\hat{X} = V_p\Lambda_p^{\frac{1}{2}}$, with $B = \hat{X}\hat{X}^T$. Clearly from the information in the dissimilarities one can only recover the original coordinates up to a translation, a rotation and reflections of the axes; the solution obtained for $\hat{X}$ is such that the origin is at the mean of the $n$ points, and that the axes chosen by the procedure are the principal axes of the $\hat{X}$ configuration.

It may not be necessary to uses all $p$ dimensions to obtain a reasonable approximation; a configuration $\hat{X}$ in $k$-dimensions can be obtained by using the largest $k$ eigenvalues so that $\hat{X} = V_k\Lambda_k^{\frac{1}{2}}$. These are known as the principal coordinates of $X$ in $k$ dimensions. The fraction of the variance explained by the first $k$ eigenvalues is $\sum_{i=1}^k \lambda_i / \sum_{i=1}^n \lambda_i$.

Classical scaling as explained above works on Euclidean distances as the dissimilarities. However, one can run the same algorithm with a non-Euclidean dissimilarity matrix, although in this case there is no guarantee that the eigenvalues will be non-negative.

Classical scaling derives from the work of Schoenberg and Young and Householder in the 1930's. Expositions of the theory can be found in [5] and [2].

### 2.1.1 Optimality properties of classical scaling

Mardia *et al* [5] (section 14.4) give the following optimality property of the classical scaling solution.

**Theorem 1** *Let $X$ denote a configuration of points in $\mathbb{R}^p$, with interpoint distances $\delta_{ij}^2 = (\mathbf{x}_i - \mathbf{x}_j)^T(\mathbf{x}_i - \mathbf{x}_j)$. Let $L$ be a $p \times p$ rotation matrix and set $L = (L_1, L_2)$, where $L_1$ is $p \times k$ for $k < p$. Let $\hat{X} = XL_1$, the projection of $X$ onto a $k$-dimensional subspace of $\mathbb{R}^p$, and let $\hat{d}_{ij}^2 = (\hat{\mathbf{x}}_i - \hat{\mathbf{x}}_j)^T(\hat{\mathbf{x}}_i - \hat{\mathbf{x}}_j)$. Amongst all projections $\hat{X} = XL_1$, the quantity $\phi = \sum_{i,j}(\delta_{ij}^2 - \hat{d}_{ij}^2)$ is minimized when $X$ is projected onto its principal coordinates in $k$ dimensions. For all $i,j$ we have $\hat{d}_{ij} \leq \delta_{ij}$. The value of $\phi$ for the principal coordinate projection is $\phi = 2n(\lambda_{k+1} + \ldots + \lambda_p)$.*

## 2.2 Relationships between classical scaling and PCA

There is a well-known relationship between PCA and classical scaling; see e.g. Cox and Cox (1994) section 2.2.7.

Principal components analysis (PCA) is concerned with the eigendecomposition of the sample covariance matrix $S = \frac{1}{n}X^THX$. It is easy to show that the eigenvalues of $nS$ are the $p$ non-zero eigenvalues of $B$. To see this note that $H^2 = H$ and thus that $nS = (HX)^T(HX)$. Let $\mathbf{v}_i$ be a unit-length eigenvector of $B$ so that $B\mathbf{v}_i = \lambda_i\mathbf{v}_i$. Premultiplying by $(HX)^T$ yields

$$(HX)^T(HX)(HX)^T\mathbf{v}_i = \lambda_i(HX)^T\mathbf{v}_i \qquad (1)$$

so we see that $\lambda_i$ is an eigenvalue of $nS$. $\mathbf{y}_i = (HX)^T\mathbf{v}_i$ is the corresponding eigenvector; note that $\mathbf{y}_i^T\mathbf{y}_i = \lambda_i$. Centering $X$ and projecting onto the unit vector $\hat{\mathbf{y}}_i = \lambda_i^{-1/2}\mathbf{y}_i$ we obtain

$$HX\hat{\mathbf{y}}_i = \lambda_i^{-1/2}HX(HX)^T\mathbf{v}_i = \lambda_i^{1/2}\mathbf{v}_i. \qquad (2)$$

Thus we see that the projection of $X$ onto the eigenvectors of $nS$ returns the classical scaling solution.

## 2.3 Metric MDS

The aim of classical scaling is to find a configuration of points $\hat{X}$ so that the interpoint distances $d_{ij}$ well approximate the dissimilarities $\delta_{ij}$. In metric MDS this criterion is relaxed, so that instead we require

$$d_{ij} \approx f(\delta_{ij}), \qquad (3)$$

where $f$ is a specified (analytic) function. For this definition see, e.g. Kruskal and Wish [4] (page 22), where polynomial transformations are suggested.

A straightforward way to carry out metric MDS is to define a error function (or stress)

$$S = \frac{\sum_{i,j} w_{ij}(d_{ij} - f(\delta_{ij}))^2}{\sum_{i,j} d_{ij}^2}, \qquad (4)$$

where the $\{w_{ij}\}$ are appropriately chosen weights. One can then obtain derivatives of $S$ with respect to the coordinates of the points that define the $d_{ij}$'s and use gradient-based (or more sophisticated methods) to minimize the stress. This method is known as least-squares scaling. An early reference to this kind of method is Sammon (1969) [6], where $w_{ij} = 1/\delta_{ij}$ and $f$ is the identity function.

Note that if $f(\delta_{ij})$ has some adjustable parameters $\theta$ and is linear with respect to $\theta$ [2], then the function $f$ can also be adapted and the optimal value for those parameters given the current $d_{ij}$'s can be obtained by (weighted) least-squares regression.

Critchley (1978) [3] (also mentioned in section 2.4.2 of Cox and Cox) carried out metric MDS by running the classical scaling algorithm on the transformed dissimilarities. Critchley suggests the power transformation $f(\delta_{ij}) = \delta_{ij}^{\mu}$ (for $\mu > 0$). If the dissimilarities are derived from Euclidean distances, we note that the kernel $k(\mathbf{x}, \mathbf{y}) = -||\mathbf{x} - \mathbf{y}||^{\beta}$ is conditionally positive definite (CPD) if $\beta \leq 2$ [1]. When the kernel is CPD, the centered matrix will be positive definite. Critchley's use of the classical scaling algorithm is similar to the algorithm discussed below, but crucially the kernel PCA method ensures that the matrix $B$ derived form the transformed dissimilarities is non-negative definite, while this is not guaranteed by Critchley's transformation for arbitrary $\mu$.

A further member of the MDS family is nonmetric MDS (NMDS), also known as ordinal scaling. Here it is only the relative rank ordering between the $d$'s and the $\delta$'s that is taken to be important; this constraint can be imposed by demanding that the function $f$ in equation 3 is monotonic. This constraint makes sense for some kinds of dissimilarity data (e.g. from psychology) where only the rank orderings have real meaning.

# 3 Kernel PCA

In recent years there has been an explosion of work on *kernel methods*. For supervised learning these include support vector machines [8], Gaussian process prediction (see, e.g. [10]) and spline methods [9]. The basic idea of these methods is to use the "kernel trick". A point $\mathbf{x}$ in the original space is re-represented as a point $\phi(\mathbf{x})$ in a $N_F$-dimensional feature space[3] $F$, where $\phi(\mathbf{x}) = (\phi_1(\mathbf{x}), \phi_2(\mathbf{x}), \ldots, \phi_{N_F}(\mathbf{x}))$. We can think of each function $\phi_j(\cdot)$ as a non-linear mapping. The key to the kernel trick is to realize that for many algorithms, the only quantities required are of the form[4] $\phi(\mathbf{x}_i).\phi(\mathbf{x}_j)$ and thus if these can be easily computed by a non-linear function $k(\mathbf{x}_i, \mathbf{x}_j) = \phi(\mathbf{x}_i).\phi(\mathbf{x}_j)$ we can save much time and effort.

Schölkopf, Smola and Müller [7] used this trick to define *kernel PCA*. One could compute the covariance matrix in the feature space and then calculate its eigenvectors/eigenvalues. However, using the relationship between $B$ and the sample covariance matrix $S$ described above, we can instead consider the $n \times n$ matrix $K$ with entries $K_{ij} = k(\mathbf{x}_i, \mathbf{x}_j)$ for $i, j = 1, \ldots, n$. If $N_F > n$ using $K$ will be more efficient than working with the covariance matrix in feature space and anyway the latter would be singular.

The data should be centered in the feature space so that $\sum_{i=1}^{n} \phi(\mathbf{x}_i) = \mathbf{0}$. This is achieved by carrying out the eigendecomposition of $\tilde{K} = HKH$ which gives the coordinates of the approximating points as described in section 2.2. Thus we see that the visualization of data by projecting it onto the first $k$ eigenvectors is exactly classical scaling in feature space.

# 4 A relationship between kernel PCA and metric MDS

We consider two cases. In section 4.1 we deal with the case that the kernel is isotropic and obtain a close relationship between kernel PCA and metric MDS. If the kernel is non-stationary a rather less close relationship is derived in section 4.2.

## 4.1 Isotropic kernels

A kernel function is stationary if $k(\mathbf{x}_i, \mathbf{x}_j)$ depends only on the vector $\boldsymbol{\tau} = \mathbf{x}_i - \mathbf{x}_j$. A stationary covariance function is isotropic if $k(\mathbf{x}_i, \mathbf{x}_j)$ depends only on the distance $\delta_{ij}$ with $\delta_{ij}^2 = \boldsymbol{\tau}.\boldsymbol{\tau}$, so that we write $k(\mathbf{x}_i, \mathbf{x}_j) = r(\delta_{ij})$. Assume that the kernel is scaled so that $r(0) = 1$. An example of an isotropic kernel is the squared exponential or RBF (radial basis function) kernel $k(\mathbf{x}_i, \mathbf{x}_j) = \exp\{-\theta(\mathbf{x}_i - \mathbf{x}_j)^T(\mathbf{x}_i - \mathbf{x}_j)\}$, for some parameter $\theta > 0$.

Consider the Euclidean distance in feature space $\tilde{\delta}_{ij}^2 = (\phi(\mathbf{x}_i) - \phi(\mathbf{x}_j))^T(\phi(\mathbf{x}_i) - \phi(\mathbf{x}_j))$. With an isotropic kernel this can be re-expressed as $\tilde{\delta}_{ij}^2 = 2(1 - r(\delta_{ij}))$. Thus the matrix $A$ has elements $a_{ij} = r(\delta_{ij}) - 1$, which can be written as $A = K - \mathbf{1}\mathbf{1}^T$. It can be easily verified that the centering matrix $H$ annihilates $\mathbf{1}\mathbf{1}^T$, so that $HAH = HKH$.

We see that the configuration of points derived from performing classical scaling on $K$ actually aims to approximate the feature-space distances computed as $\tilde{\delta}_{ij} = \sqrt{2(1 - r(\delta_{ij}))}$. As the $\tilde{\delta}_{ij}$'s are a non-linear function of the $\delta_{ij}$'s this procedure (kernel MDS) is an example of metric MDS.

**Remark 1** Kernel functions are usually chosen to be conditionally positive definite, so that the eigenvalues of the matrix $\tilde{K}$ will be non-negative. Choosing arbitrary functions to transform the dissimilarities will not give this guarantee.

**Remark 2** In nonmetric MDS we require that $d_{ij} \approx f(\delta_{ij})$ for some *monotonic* function $f$. If the kernel function $r$ is monotonically decreasing then clearly $1 - r$ is monotonically increasing. However, there are valid isotropic kernel (covariance) functions which are non-monotonic (e.g. the exponentially damped cosine $r(\delta) = e^{-\theta\delta}\cos(\omega\delta)$; see [11] for details) and thus we see that $f$ need not be monotonic in kernel MDS.

**Remark 3** One advantage of PCA is that it defines a mapping from the original space to the principal coordinates, and hence that if a new point $\mathbf{x}$ arrives, its projection onto the principal coordinates defined by the original $n$ data points can be computed[5]. The same property holds in kernel PCA, so that the computation of the projection of $\phi(\mathbf{x})$ onto the $r$th principal direction in feature space can be computed using the kernel trick as $\sum_{i=1}^{n} \alpha_i^r k(\mathbf{x}, \mathbf{x}_i)$, where $\boldsymbol{\alpha}^r$ is the $r$th eigenvector of $\tilde{K}$ (see equation 4.1 in [7]). This projection property does not hold for algorithms that simply minimize the stress objective function; for example the Sammon "mapping" algorithm [6] does not in fact define a mapping.

## 4.2 Non-stationary kernels

Sometimes non-stationary kernels (e.g. $k(\mathbf{x}_i, \mathbf{x}_j) = (1 + \mathbf{x}_i.\mathbf{x}_j)^m$ for integer $m$) are used. For non-stationary kernels we proceed as before and construct $\tilde{\delta}_{ij}^2 = (\phi(\mathbf{x}_i) - \phi(\mathbf{x}_j))^T(\phi(\mathbf{x}_i) - \phi(\mathbf{x}_j))$. We can again show that the kernel MDS procedure operates on the matrix $HKH$. However, the distance $\tilde{\delta}_{ij}$ in feature space is not a function of $\delta_{ij}$ and so the relationship of equation 3 does not hold. The situation can be saved somewhat if we follow Mardia *et al* (section 14.2.3) and relate similarities

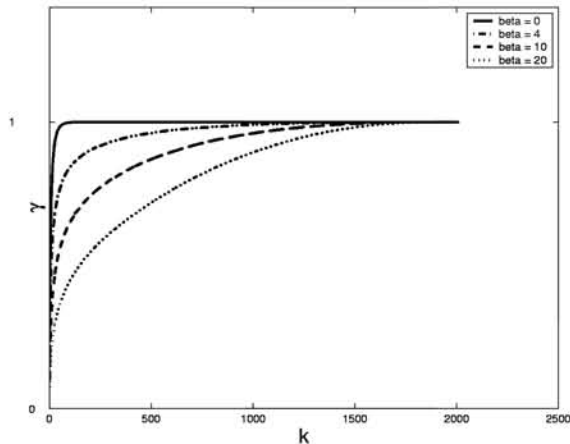

Figure 1: The plot shows $\gamma$ as a function of $k$ for various values of $\beta = \theta/256$ for the USPS test set.

to dissimilarities through $\tilde{\delta}_{ij}^2 = \tilde{c}_{ii} + \tilde{c}_{jj} - 2\tilde{c}_{ij}$, where $\tilde{c}_{ij}$ denotes the similarity between items $i$ and $j$ in feature space. Then we see that the similarity in feature space is given by $\tilde{c}_{ij} = \phi(\mathbf{x}_i).\phi(\mathbf{x}_j) = k(\mathbf{x}_i, \mathbf{x}_j)$. For kernels (such as polynomial kernels) that are functions of $\mathbf{x}_i.\mathbf{x}_j$ (the similarity in input space), we see then that the similarity in feature space is a non-linear function of the similarity measured in input space.

## 5    Choice of kernel

Having performed kernel MDS one can plot the scatter diagram (or Shepard diagram) of the dissimilarities against the fitted distances. We know that for each pair the fitted distance $d_{ij} \le \tilde{\delta}_{ij}$ because of the projection property in feature space. The sum of the residuals is given by $2n \sum_{i=k+1}^{n} \lambda_i$ where the $\{\lambda_i\}$ are the eigenvalues of $\tilde{K} = HKH$. (See Theorem 1 above and recall that at most $n$ of the eigenvalues of the covariance matrix in feature space will be non-zero.) Hence the fraction of the sum-squared distance explained by the first $k$ dimensions is $\gamma = \sum_{i=1}^{k} \lambda_i / \sum_{i=1}^{n} \lambda_i$.

One idea for choosing the kernel would be to fix the dimensionality $k$ and choose $r(\cdot)$ so that $\gamma$ is maximized. Consider the effect of varying $\theta$ in the RBF kernel

$$k(\mathbf{x}_i, \mathbf{x}_j) = \exp\{-\theta(\mathbf{x}_i - \mathbf{x}_j)^T(\mathbf{x}_i - \mathbf{x}_j)\}. \tag{5}$$

As $\theta \to \infty$ we have $\tilde{\delta}_{ij}^2 = 2(1 - \delta(i, j))$ (where $\delta(i, j)$ is the Kronecker delta), which are the distances corresponding to a regular simplex. Thus $K \to I_n$, $HKH = H$ and $\gamma = k/(n-1)$. Letting $\theta \to 0$ and using $e^{-\theta z} \simeq 1 - \theta z$ for small $\theta$, we can show that $K_{ij} = 1 - \theta \delta_{ij}^2$ as $\theta \to 0$, and thus that the classical scaling solution is obtained in this limit.

Experiments have been run on the US Postal Service database of handwritten digits, as used in [7]. The test set of 2007 images was used. The size of each image is $16 \times 16$ pixels, with the intensity of the pixels scaled so that the average variance over all 256 dimensions is 0.5. In Figure 1 $\gamma$ is plotted against $k$ for various values of $\beta = \theta/256$. By choosing an index $k$ one can observe from Figure 1 what fraction of the variance is explained by the first $k$ eigenvalues. The trend is that as $\theta$ decreases more and

more variance is explained by fewer components, which fits in with the idea above that the $\theta \to \infty$ limit gives rise to the regular simplex case. Thus there does not seem to be a non-trivial value of $\theta$ which minimizes the residuals.

## 6   Discussion

The results above show that kernel PCA using an isotropic kernel function can be interpreted as performing a kind of metric MDS. The main difference between the kernel MDS algorithm and other metric MDS algorithms is that kernel MDS uses the classical scaling solution in feature space. The advantage of the classical scaling solution is that it is computed from an eigenproblem, and avoids the iterative optimization of the stress objective function that is used for most other MDS solutions. The classical scaling solution is unique up to the unavoidable translation, rotation and reflection symmetries (assuming that there are no repeated eigenvalues). Critchley's work (1978) is somewhat similar to kernel MDS, but it lacks the notion of a projection into feature space and does not always ensure that the matrix $B$ is non-negative definite.

We have also looked at the question of adapting the kernel so as to minimize the sum of the residuals. However, for the case investigated this leads to a trivial solution.

### Acknowledgements

I thank David Willshaw, Matthias Seeger and Amos Storkey for helpful conversations, and the anonymous referees whose comments have helped improve the paper.

## Footnotes

[1]In fact if the points are not in "general position" the number of zero eigenvalues will be greater than $n - p$. Below we assume that the points are in general position, although the arguments can easily be carried through with minor modifications if this is not the case.

[2] $f$ can still be a non-linear function of its argument.

[3]For some kernels $N_F = \infty$.

[4]We denote the inner product of two vectors as either $\mathbf{a}.\mathbf{b}$ or $\mathbf{a}^T\mathbf{b}$.

[5]Note that this will be, in general, different to the solution found by doing PCA on the full data set of $n + 1$ points.

## References

[1] C. Berg, J. P. R. Christensen, and P. Ressel. *Harmonic Analysis on Semigroups.* Springer-Verlag, New York, 1984.

[2] T. F. Cox and M. A. A. Cox. *Multidimensional Scaling.* Chapman and Hall, London, 1994.

[3] F. Critchley. Multidimensionsal scaling: a short critique and a new method. In L. C. A Corsten and J. Hermans, editors, *COMPSTAT 1978.* Physica-Verlag, Vienna, 1978.

[4] J. B. Kruskal and M. Wish. *Multidimensional Scaling.* Sage Publications, Beverly Hills, 1978.

[5] Mardia, K. V. and Kent, J. T. and Bibby, J. M. *Multivariate Analysis.* Academic Press, 1979.

[6] J. W. Sammon. A nonlinear mapping for data structure analysis. *IEEE Trans. on Computers,* 18:401–409, 1969.

[7] B. Schölkopf, A. Smola, and K.-R. Müller. Nonlinear component analysis as a kernel eigenvalue problem. *Neural Computation,* 10:1299–1319, 1998.

[8] V. N. Vapnik. *The nature of statistical learning theory.* Springer Verlag, New York, 1995.

[9] G. Wahba. *Spline models for observational data.* Society for Industrial and Applied Mathematics, Philadelphia, PA, 1990. CBMS-NSF Regional Conference series in applied mathematics.

[10] C. K. I. Williams and D. Barber. Bayesian classification with Gaussian processes. *IEEE Transactions on Pattern Analysis and Machine Intelligence,* 20(12):1342–1351, 1998.

[11] A. M. Yaglom. *Correlation Theory of Stationary and Related Random Functions Volume I:Basic Results.* Springer Verlag, 1987.
